# On Causal Discovery with Cyclic Additive Noise Models

**Joris M. Mooij**
Radboud University
Nijmegen, The Netherlands
j.mooij@cs.ru.nl

**Dominik Janzing**
Max Planck Institute for Intelligent Systems
Tübingen, Germany
dominik.janzing@tuebingen.mpg.de

**Tom Heskes**
Radboud University
Nijmegen, The Netherlands
t.heskes@cs.ru.nl

**Bernhard Schölkopf**
Max Planck Institute for Intelligent Systems
Tübingen, Germany
bs@tuebingen.mpg.de

## Abstract

We study a particular class of cyclic causal models, where each variable is a (possibly nonlinear) function of its parents and additive noise. We prove that the causal graph of such models is generically identifiable in the bivariate, Gaussian-noise case. We also propose a method to learn such models from observational data. In the acyclic case, the method reduces to ordinary regression, but in the more challenging cyclic case, an additional term arises in the loss function, which makes it a special case of nonlinear independent component analysis. We illustrate the proposed method on synthetic data.

## 1 Introduction

Causal discovery refers to a special class of statistical and machine learning methods that infer causal relationships between variables from data and prior knowledge [1, 2, 3]. Whereas in machine learning, one traditionally concentrates on the task of predicting the values of variables given *observations* of other variables (for example in regression or classification tasks), causal discovery focuses on predicting the results of *interventions* on the system: if one forces one (or more) of the variables into a particular state, how will the probability distribution of the other variables be affected? In this sense, causal discovery concentrates more on inferring the underlying *mechanism* that generated the data than on modeling the data itself.

An important assumption often made in causal discovery is that the causal mechanism is *acyclic*, i.e., that no feedback loops are present in the system. For example, if A causes B, and B causes C, then the possibility that C also causes A is usually excluded from the outset. This acyclicity assumption is useful because it simplifies the theoretical analysis and often is also a reasonable assumption to make. Nevertheless, causal cycles are known to occur frequently in biological systems such as gene regulatory networks and protein interaction networks. One would expect that taking such feedback loops into account during data analysis should therefore significantly improve the quality of the inferred causal structure.

Essentially two strategies for dealing with cycles in causal models can be distinguished. The first one is to perform repeated measurements in time, and to infer a causal model for the dynamics of the underlying system. The fact that causes always precede their effects provides additional prior knowledge that simplifies causal discovery, which is exploited in methods based on Granger causality [4]. Additionally, under certain assumptions, "unrolling" the model in time effectively removes the cycles, which is used in methods such as vector auto-regressive models, which are popular in

econometrics, or more generally, Dynamic Bayesian Networks [5] and ordinary differential equation models. However, all these methods need time series data where the temporal resolution of the measurements is high relative to the characteristic time scale of the feedback loops in order to rule out instantaneous cyclic relationships. Therefore, a significant practical drawback of this strategy is that obtaining time series data with sufficiently high temporal resolution is often costly—or even impossible—using current technology.

The second strategy is based on the assumption that the system is in equilibrium, and that the data have been gathered from an equilibrium distribution (in the ergodic case, the data can also consist of snapshots of the dynamical system, taken at different points in time). The equilibrium distribution is then used to draw conclusions about the underlying dynamic system, and to predict the results of interventions. This is the approach taken in the current paper. We assume the equilibrium to be described by fixed point equations, where each variable is a function of some other variables, plus noise. This noise models unobserved causes and is assumed to be different for each independent realization of the system, but constant during equilibration. In the simplest case (assuming causal sufficiency), the noise terms are jointly independent. Together, these assumptions define an interesting model class that forms a direct generalization of Structural Equation Models (SEMs) [2] to the nonlinear (and cyclic) case.

An important novel aspect of our work is that we consider continuous-valued variables and *nonlinear* causal mechanisms. Although the linear case has been studied in considerable detail already [6, 7, 8], as far as we know, nobody has yet investigated the (more realistic) case of nonlinear causal mechanisms. The basic assumption made in [7] is the so-called *Global Directed Markov Condition*, which relates (conditional) independences between the variables with the structure of the causal graph. In the cyclic case, however, it is not obvious what the relationship is with the class of nonlinear causal models that we consider here. Therefore, direct generalization of the algorithm proposed in [7] to the nonlinear case seems difficult. Furthermore, conditional independences only allow identification of the graph up to Markov equivalence classes. For instance, in the bivariate case, one cannot distinguish between $X \to Y$, $Y \to X$ and $X \leftrightarrows Y$ using conditional independences alone. Researchers have also studied cyclic causal models with discrete variables [9, 10]. However, if the measured variables are intrinsically continuous-valued, it is desirable to avoid discretization as a preprocessing step, as this throws away information that is useful for causal discovery.

## 2 Cyclic additive noise models

Let $V$ be a finite index set. Let $(X_i)_{i \in V}$ be random variables modeling measurable properties of the system of interest and let $(E_i)_{i \in V}$ be other random variables modeling unobservable noise sources. We assume that all random variables take values in the real numbers. We also assume that the noise variables $(E_i)_{i \in V}$ have densities and are jointly independent:

$$p(e_V) = \prod_{i \in V} p_{E_i}(e_i). \tag{1}$$

For each $i$, let $\mathrm{pa}(i) \subseteq V \setminus \{i\}$ be a set defining the *parents of $i$* and $f_i : \mathbb{R}^{|\mathrm{pa}(i)|} \to \mathbb{R}$ be a continuously differentiable function. Under certain assumptions (see below), the following equations specify a unique probability distribution on the observable variables $(X_i)_{i \in V}$:

$$X_i = f_i(X_{\mathrm{pa}(i)}) + E_i, \qquad i \in V. \tag{2}$$

Using vector notation, we can write the fixed point equations (2) in a more compact manner as

$$\boldsymbol{X} = \boldsymbol{f}(\boldsymbol{X}) + \boldsymbol{E}. \tag{3}$$

The probability distribution $p(\boldsymbol{X})$ induced by these equations is interpreted as the *equilibrium* distribution of an underlying dynamic system. Each function $f_i$ represents a *causal mechanism* which determines $X_i$ as a function of its parents $X_{\mathrm{pa}(i)}$, which model its *direct causes*. The noise variables can be interpreted as other, unobserved causes for their corresponding variables. By assuming independence of the noise variables, we are assuming *causal sufficiency*, or in other words, absence of confounders (hidden common causes).

We call a model specified by (1) and (2) an *additive noise model*. With any additive noise model we can associate a directed graph with vertices $V$ and directed edges $i \to j$ if $i \in \mathrm{pa}(j)$, i.e., from

causes to their direct effects.[1] If this graph is acyclic, we call the model an *acyclic additive noise model*. If the graph contains (directed) cycles, we call the model a *cyclic additive noise model*.[2]

**Interpretation in the cyclic case**
Note that the presence of cycles increases the complexity of the model, because the equations (2) become recursive. The interpretation of these equations also becomes less straightforward in the cyclic case. In general, for a fixed noise value $\boldsymbol{E} = \boldsymbol{e}$, the fixed point equations $\boldsymbol{x} = \boldsymbol{f}(\boldsymbol{x}) + \boldsymbol{e}$ can have any number of fixed points between 0 and $\infty$. For simplicity, however, we will assume that for each noise value $\boldsymbol{e}$ there exists a unique fixed point $\boldsymbol{x} = \boldsymbol{F}(\boldsymbol{e})$. Later, in Section 3.1, we will give a sufficient condition for this to be the case. Under this assumption, the joint probability distribution $p(\boldsymbol{E})$ induces a unique joint probability distribution $p(\boldsymbol{X})$.

This interpretation also shows a way to sample from the joint distribution: First, one samples a joint value of the noise $\boldsymbol{e}$. Then, one iterates the fixed point equations (2) to find the corresponding fixed point $\boldsymbol{x} = \boldsymbol{F}(\boldsymbol{e})$. This yields one sample $\boldsymbol{x}$. Different independent samples are obtained by repeating this process. Thus, the equations can be interpreted as the equilibrium distribution of a dynamic system in the presence of noise which is constant during equilibration, but differs across measurements (data points). If in reality the noise does change over time, but on a slow time scale relative to the time scale of the equilibration, then this model can be considered as the first-order approximation.

**The induced density**
Although the mapping $\boldsymbol{F} : \boldsymbol{e} \mapsto \boldsymbol{x}$ that maps noise values to their corresponding fixed points under (3) is nontrivial in most cases, a crucial observation is that its inverse $\boldsymbol{G} = \boldsymbol{F}^{-1} = \boldsymbol{I} - \boldsymbol{f}$ has a very simple form (here, $\boldsymbol{I}$ is the identity mapping). Under the change of variables $\boldsymbol{e} \mapsto \boldsymbol{x}$, the transformation rule of the densities reads:

$$p_{\boldsymbol{X}}(\boldsymbol{x}) = p_{\boldsymbol{E}}\big(\boldsymbol{x} - \boldsymbol{f}(\boldsymbol{x})\big) \, |\boldsymbol{I} - \nabla \boldsymbol{f}(\boldsymbol{x})| = |\boldsymbol{I} - \nabla \boldsymbol{f}(\boldsymbol{x})| \prod_{i \in V} p_{E_i}\big(x_i - f_i(\boldsymbol{x}_{\mathrm{pa}(i)})\big) \qquad (4)$$

where $\nabla \boldsymbol{f}(\boldsymbol{x})$ is the Jacobian of $\boldsymbol{f}$ evaluated at $\boldsymbol{x}$ and $|\cdot|$ denotes the absolute value of the determinant of a matrix.

Note that although *sampling* from the distribution $p_{\boldsymbol{X}}$ is elaborate (as it typically involves many iterations of the fixed point equations), the corresponding density can be easily expressed analytically in terms of the noise distributions and partial derivatives of the causal mechanisms. Later we will see that the fact that the model has a simple structure in the "backwards" direction allows us to efficiently learn it from data, which may be surprising considering the fact that the model is complex in the "forward" direction.

**Causal interpretation**
An additive noise model can be used for ordinary prediction tasks (i.e., predict some of the variables conditioned on observations of some other variables), but can also be used to predict the results of *interventions*: if we force some of the variables to certain values, what will happen with the others? Such an intervention can be modeled by replacing the equations for the intervened variables by simple equations $X_i = C_i$, with $C_i$ the value set by the intervention. This procedure results in another additive noise model. If the altered fixed point equations induce a unique probability distribution on $\boldsymbol{X}$, then this is the predicted distribution on $\boldsymbol{X}$ under the intervention. In this sense, additive noise models are given a *causal* interpretation. Hereafter, we will therefore refer to the graph associated with the additive noise model as the *causal* graph.

## 3 Identifiability

An interesting and important question for causal discovery is under which conditions the causal graph is identifiable given only the joint distribution $p(\boldsymbol{X})$. Lacerda *et al.* [8] have shown that under

the additional assumption of linearity (i.e., all functions $f_i$ are linear), the causal graph is completely identifiable if at most one of the noise sources has a Gaussian distribution. The proof is based on Independent Component Analysis. Our aim here is to deal with the more difficult nonlinear case. In this work, we focus our attention on the bivariate case. Our main result, Theorem 1, can be seen as an extension of the identifiability result for acyclic nonlinear additive noise models derived in [11], although we make the additional simplifying assumption that the noise variables are Gaussian. We believe that similar identifiability results can be derived in the multivariate case ($|V| > 2$) and for non-Gaussian noise distributions. However, proving such results seems to be significantly harder as the calculations become very cumbersome, and we leave this as an open problem for future work.

## 3.1 The bivariate case

Before we state our identifiability result, we first give a sufficient condition for existence of a unique equilibrium distribution for the bivariate case.

**Lemma 1** *Consider the fixed point equations $x = f_X(y) + c_X$, $y = f_Y(x) + c_Y$ parameterized by constants $(c_X, c_Y)$. If $\sup_{x,y} |f'_X(y) f'_Y(x)| = r < 1$, then for any $(c_X, c_Y)$, the fixed point equations converge to a unique fixed point that does not depend on the initial conditions.*

**Proof.** Consider the mapping defined by applying the fixed point equations twice. Its Jacobian is diagonal and the absolute values of the entries are bounded from above by $r < 1$ under the assumption above. According to Banach's fixed point theorem, it is a contraction (e.g., with respect to the Euclidean norm on $\mathbb{R}^2$) and therefore has a fixed point that is unique. Independent of the initial conditions, under repeated application of this mapping, one converges to this fixed point. Lemma 1 in the supplement then shows that the same conclusion must hold for the mapping that applies the fixed point equations only once. $\square$

This lemma provides a sufficient condition for an additive noise model to be well-defined in the bivariate case. Also, the result of any intervention will be well-defined under this condition.

Now suppose we are given the joint distribution $p_{X,Y}$ of two real-valued random variables $X, Y$ which is induced by an additive noise model. The question is whether we can identify the causal graph corresponding with the true model out of the four possibilities ($X \quad Y$, $X \rightarrow Y$, $Y \rightarrow X$, $X \leftrightarrows Y$). Hoyer *et al.* [11] have shown that if one excludes the cyclic case $X \leftrightarrows Y$, then in the generic case, the causal structure is identifiable. Our aim is to prove a stronger identifiability result where the cyclic case is not excluded *a priori*. As a first step in this direction, we consider here the case of Gaussian noise.

**Theorem 1** *Let $p_{X,Y}$ be induced by two additive Gaussian noise models, $\mathcal{M}$ and $\tilde{\mathcal{M}}$:*

$$X = f_X(Y) + E_X, Y = f_Y(X) + E_Y, E_X \perp\!\!\!\perp E_Y, E_X \sim \mathcal{N}(0, \alpha_X^{-1}), E_Y \sim \mathcal{N}(0, \alpha_Y^{-1}) \quad (\mathcal{M})$$

$$X = \tilde{f}_X(Y) + \tilde{E}_X, Y = \tilde{f}_Y(X) + \tilde{E}_Y, \tilde{E}_X \perp\!\!\!\perp \tilde{E}_Y, \tilde{E}_X \sim \mathcal{N}(0, \tilde{\alpha}_X^{-1}), \tilde{E}_Y \sim \mathcal{N}(0, \tilde{\alpha}_Y^{-1}) \quad (\tilde{\mathcal{M}})$$

*Assuming that $\sup_{x,y} |f'_X(y) f'_Y(x)| < 1$ and similarly $\sup_{x,y} \left| \tilde{f}'_X(y) \tilde{f}'_Y(x) \right| < 1$, then the two corresponding causal graphs coincide: $\mathcal{G}_\mathcal{M} = \mathcal{G}_{\tilde{\mathcal{M}}}$, i.e.:*

$$f_X \text{ is constant} \iff \tilde{f}_X \text{ is constant}, \quad \text{and} \quad f_Y \text{ is constant} \iff \tilde{f}_Y \text{ is constant},$$

*or the models are of the following very special form:*

- *either: $f_X$, $\tilde{f}_X$, $f_Y$, $\tilde{f}_Y$ are all affine,*
- *or: one model (say $\tilde{\mathcal{M}}$) is acyclic, the other is cyclic, and the following equations hold:*

$$f_Y(x) = Cx + D \text{ with } C \neq 0, f_X(y) = \frac{\tilde{\alpha}_X}{\alpha_X} \tilde{f}_X(y) - \frac{\alpha_Y}{\alpha_X} Cy + \frac{\alpha_Y}{\alpha_X} CD, \tilde{f}_Y(x) = \tilde{D} \quad (5)$$

*and $\tilde{f}_X$ satisfies the following differential equation:[3]*

$$-\frac{1}{\alpha_X}(\tilde{\alpha}_X \tilde{f}_X - \alpha_Y C y + \alpha_Y C D)(\tilde{\alpha}_X \tilde{f}'_X - \alpha_Y C) + \tilde{\alpha}_X \tilde{f}_X \tilde{f}'_X$$

$$= \alpha_Y(y - D) - \tilde{\alpha}_Y(y - \tilde{D}) + C \frac{\tilde{\alpha}_X \tilde{f}''_X}{\alpha_X - (\tilde{\alpha}_X \tilde{f}'_X - \alpha_Y C) C}. \tag{6}$$

We will only sketch the proof here, and refer to the supplementary material for the details. What the theorem shows is that, apart from a small class of exceptions, bivariate additive Gaussian-noise models induce densities that allow a perfect reconstruction of the causal graph. In a certain sense, the situation can be seen as similar to the well-known "faithfulness assumption" [3]: the latter assumption is often made in order to exclude the highly special cases of causal models which would spoil identifiability of the Markov equivalence class. The usual reasoning is that these cases are so rare that they can be ignored in practice. A similar reasoning can be made in our case.

Although our main identifiability result, Theorem 1, may seem rather restricted as it only considers two variables, it may be possible to use this two-variable identifiability result as a key building block for deriving more general identifiability results for many variables, similar as how [12] generalized the (acyclic) identifiability result of [11] from two to many variables.

## 3.2 Proof sketch

Writing $\pi_{...}(\cdots) := \log p_{...}(\cdots)$ for logarithms of densities, we reexpress (4) for the bivariate case:

$$\pi_{X,Y}(x,y) = \pi_{E_X}\big(x - f_X(y)\big) + \pi_{E_Y}\big(y - f_Y(x)\big) + \log|1 - f_X'(y)f_Y'(x)| \tag{7}$$

Partial differentiation with respect to $x$ and $y$ yields the following equation, which will be the equation on which we base our identifiability proof:

$$\frac{\partial^2 \pi_{X,Y}}{\partial x \partial y} = -\pi_{E_X}''\big(x - f_X(y)\big)f_X'(y) - \pi_{E_Y}''\big(y - f_Y(x)\big)f_Y'(x) - \frac{f_X''(y)f_Y''(x)}{\big(1 - f_X'(y)f_Y'(x)\big)^2} \tag{8}$$

We will now specialize to Gaussian noise and give a sketch of how to prove identifiability of the causal graph. We assume $E_X \sim \mathcal{N}(0, \alpha_X^{-1})$ and $E_Y \sim \mathcal{N}(0, \alpha_Y^{-1})$ where $\alpha_X = \sigma_X^{-2}, \alpha_Y = \sigma_Y^{-2}$ are the precisions (inverse variances) of the Gaussian noise variables. Equation (8) simplifies to:

$$\frac{\partial^2 \pi_{X,Y}}{\partial x \partial y} = \alpha_X f_X'(y) + \alpha_Y f_Y'(x) - \frac{f_X''(y)f_Y''(x)}{\big(1 - f_X'(y)f_Y'(x)\big)^2} \tag{9}$$

A similar equation holds for the other model:

$$\frac{\partial^2 \pi_{X,Y}}{\partial x \partial y} = \tilde{\alpha}_X \tilde{f}_X'(y) + \tilde{\alpha}_Y \tilde{f}_Y'(x) - \frac{\tilde{f}_X''(y)\tilde{f}_Y''(x)}{\big(1 - \tilde{f}_X'(y)\tilde{f}_Y'(x)\big)^2} \tag{10}$$

The general idea of the identifiability proof is as follows. We consider two cases: (i) model $\tilde{\mathcal{M}}$ has zero "arrows", i.e., $\tilde{f}_X' = 0$ and $\tilde{f}_Y' = 0$; (ii) model $\tilde{\mathcal{M}}$ has one "arrow", say, $\tilde{f}_X' \neq 0, \tilde{f}_Y' = 0$. By equating the r.h.s.'s of (9) and (10), we show in both cases that generically (i.e., except for very special choices of the model parameters), model $\mathcal{M}$ must equal model $\tilde{\mathcal{M}}$. This then implies that the causal graphs of $\mathcal{M}$ and $\tilde{\mathcal{M}}$ must be the same in the generic case.

For example, in the first case, because $\tilde{f}_X' = \tilde{f}_Y' = 0$, we obtain the following equation:

$$0 = \big(\alpha_X f_X'(y) + \alpha_Y f_Y'(x)\big)\big(1 - f_X'(y)f_Y'(x)\big)^2 - f_X''(y)f_Y''(x) \tag{11}$$

This is a nonlinear partial differential equation in $\phi(x) := f_Y'(x)$ and $\psi(y) := f_X'(y)$. Inspired by the identifiability proof in [13], we adopt the solution method from [14, Supplement S.4.3] that gives a general method for solving functional-differential equations of the form

$$\Phi_1(x)\Psi_1(y) + \Phi_2(x)\Psi_2(y) + \cdots + \Phi_k(x)\Psi_k(y) = 0 \tag{12}$$

where the functionals $\Phi_i(x)$ and $\Psi_i(y)$ depend only on $x$ and $y$, respectively:

$$\Phi_i(x) = \Phi_i(x, \phi, \phi'), \qquad \Psi_i(y) = \Psi_i(y, \psi, \psi').$$

The idea behind the solution method is to repeatedly divide by one of the functionals and differentiate with respect to the corresponding variable. For example, dividing by $\Phi_1$ and differentiating with respect to $x$, we obtain:

$$\left(\frac{\partial}{\partial x}\frac{\Phi_2(x)}{\Phi_1(x)}\right)\Psi_2(y) + \cdots + \left(\frac{\partial}{\partial x}\frac{\Phi_k(x)}{\Phi_1(x)}\right)\Psi_k(y) = 0$$

which is again of the form (12), but with one fewer term. This process is repeated until an equation of the form (12) remains with only 2 terms. That equation is easily solved, as its general solution can be written as

$$C_1 \Phi_1(x) + C_2 \Phi_2(x) = 0, C_2 \Psi_1(y) - C_1 \Psi_2(y) = 0$$

for arbitrary constants $C_1, C_2 \in \mathbb{R}$, and there are also two degenerate solutions $\Phi_1 = \Phi_2 = 0$ (and $\Psi_1, \Psi_2$ arbitrary) and $\Psi_1 = \Psi_2 = 0$ (and $\Phi_1, \Phi_2$ arbitrary). These equations, which are now ordinary differential equations, can be solved by standard methods. The solutions are then substituted into the original equation (12) in order to remove redundant constants of integration. Applying this method to the case at hand, one obtains equations for $f_X'$ and $f_Y'$. Solving these equations, one finds that either $\mathcal{M} = \tilde{\mathcal{M}}$, or that $f_X' = f_Y' = \tilde{f}_X' = \tilde{f}_Y' = 0$. In the second case (where $\tilde{\mathcal{M}}$ has one arrow) the equations show that either $\mathcal{M} = \tilde{\mathcal{M}}$, or the model parameters should satisfy equations (5) and (6).

# 4 Learning additive noise models from observational data

In this section, we propose a method to learn an additive noise model from a finite data set $\mathcal{D} := \{\boldsymbol{x}^{(n)}\}_{n=1}^N$. We will only describe the bivariate case in detail, although the method can be extended to more than two variables in a straightforward way.

We first consider how we can learn the causal mechanisms $\{f_i\}_{i \in V}$ for a fixed causal structure. This can be done efficiently by a MAP estimate with respect to (the parameters of) the causal mechanisms. Using (4), the MAP problem can be written as:

$$\underset{\hat{\boldsymbol{f}}}{\arg\max}\, p(\hat{\boldsymbol{f}}) \prod_{n=1}^N \left( \left| \boldsymbol{I} - \nabla \hat{\boldsymbol{f}}(\boldsymbol{x}^{(n)}) \right| \prod_{i \in V} p_{E_i}\left( x_i^{(n)} - \hat{f}_i(\boldsymbol{x}_{\mathrm{pa}(i)}^{(n)}) \right) \right) \tag{13}$$

where $p(\hat{\boldsymbol{f}})$ specifies the prior distribution of the causal mechanisms. Note the presence of the determinant; in the acyclic case, this term becomes 1, and the method reduces to standard regression. In the cyclic case, however, the determinant is necessary in order to penalize dependencies between the estimated noise variables. One can consider this as a special case of nonlinear independent component analysis, as the MAP estimate (13) can also be interpreted as the minimizer of the mutual information between the noise variables. If the estimated functions lead to noise estimates $\hat{E}_i = X_i - \hat{f}_i(\boldsymbol{X}_{\mathrm{pa}(i)})$ which are mutually independent according to some independence test, then we accept the model.

One can try all possible causal graph structures and test which ones fit the data. The models that lead to independent estimated noise values are possible causal explanations of the data. If multiple models with different causal graphs lead to independent estimated noise values, we prefer models with fewer arrows in the graph.[4] If the number of data points is large enough, Theorem 1 suggests that for two variables with Gaussian noise, in the generic case, a unique causal structure will be identified in this way. For more than two variables, and for other noise distributions, the method can still be applied, but we do not know whether (in general and asymptotically) there will be a unique causal structure that explains the data.

We now work out the bivariate Gaussian case in more detail. The prior for the functions $\hat{\boldsymbol{f}}$ can be chosen arbitrarily, for example using some parametric approach. Here, we will use a nonparametric approach using Gaussian processes. The negative log-likelihood $\mathcal{L} := -\ln p(\mathcal{D} \mid \hat{f}_X, \hat{f}_Y)$ can be written in terms of the observational data $\mathcal{D} := \{(x^{(n)}, y^{(n)})\}_{n=1}^N$ as:

$$\mathcal{L} = -\sum_{i=1}^N \pi_{E_Y}\left(y^{(i)} - \hat{f}_Y(x^{(i)})\right) - \sum_{i=1}^N \pi_{E_X}\left(x^{(i)} - \hat{f}_X(y^{(i)})\right) - \sum_{i=1}^N \log\left|1 - \hat{f}_Y'(x^{(i)})\hat{f}_X'(y^{(i)})\right|.$$

Assuming Gaussian noise $E_X \sim \mathcal{N}(0, \sigma_X^2)$, $E_Y \sim \mathcal{N}(0, \sigma_Y^2)$ and using Gaussian Process priors for the causal mechanisms $f_X$ and $f_Y$, i.e., taking $\hat{\boldsymbol{x}} := f_X(\boldsymbol{y}) \sim \mathcal{N}\left(0, \boldsymbol{K}_X(\boldsymbol{y})\right)$ and

$\hat{\boldsymbol{y}} := f_Y(\boldsymbol{x}) \sim \mathcal{N}\big(0, \boldsymbol{K}_Y(\boldsymbol{x})\big)$ where $\boldsymbol{K}_X$ is the Gram matrix with entries $K_{X;ij} = k_X(y^{(i)}, y^{(j)})$ for some covariance function $k_X : \mathbb{R}^2 \to \mathbb{R}$, and similarly for $\boldsymbol{K}_Y$, we obtain:

$$\min_{\hat{\boldsymbol{x}}, \hat{\boldsymbol{y}}} \mathcal{L} = N \log \sigma_X + N \log \sigma_Y + \frac{1}{2} \log |\boldsymbol{K}_X| + \frac{1}{2} \log |\boldsymbol{K}_Y|$$

$$+ \min_{\hat{\boldsymbol{x}}, \hat{\boldsymbol{y}}} \left( \frac{1}{2\sigma_Y^2} \|\boldsymbol{y} - \hat{\boldsymbol{y}}\|^2 + \frac{1}{2\sigma_X^2} \|\boldsymbol{x} - \hat{\boldsymbol{x}}\|^2 + \frac{1}{2} \hat{\boldsymbol{x}}^T \boldsymbol{K}_X^{-1} \hat{\boldsymbol{x}} + \frac{1}{2} \hat{\boldsymbol{y}}^T \boldsymbol{K}_Y^{-1} \hat{\boldsymbol{y}} \right.$$

$$\left. - \sum_{i=1}^{N} \log \left| 1 - \left( \frac{\partial k_Y}{\partial x}(x^{(i)}, \boldsymbol{x}) \boldsymbol{K}_Y^{-1} \hat{\boldsymbol{y}} \right) \left( \frac{\partial k_X}{\partial y}(y^{(i)}, \boldsymbol{y}) \boldsymbol{K}_X^{-1} \hat{\boldsymbol{x}} \right) \right| \right),$$

where we used the expected derivatives of the Gaussian Processes for approximating the determinant-term. In our experiments, we used Gaussian covariance kernels

$$k_X(y, y') = \lambda_X^2 \exp \left( -\frac{(y - y')^2}{2\kappa_X^2} \right) + \rho \delta_{y, y'},$$

and likewise for $k_Y$. Note that we added a small constant ($\rho = 10^{-4}$) to the diagonal to allow for small, independent measurement errors or rounding errors (which occur because the Gram matrices are very ill-conditioned). The optimization problem can be solved numerically, e.g., using standard methods such as conjugate gradient or L-BFGS. We optimize simultaneously with respect to the noise values $\hat{\boldsymbol{x}}, \hat{\boldsymbol{y}}$ and the hyperparameters $\log \sigma_X, \log \kappa_X, \log \lambda_X, \log \sigma_Y, \log \kappa_Y, \log \lambda_Y$.

## 5 Experiments

We illustrate the method on several synthetic data sets in Figure 1. Each row shows a data set with $N = 500$ data points. Because of space constraints, we only show the learned *cyclic* additive noise models, omitting the acyclic ones. In each case, we calculated the $p$-value for independence of the two noise variables using the HSIC (Hilbert-Schmidt Independence Criterion) test [15]; for $p$-values substantially above 0 (say larger than 1%), we do not reject the null hypothesis of independence and hence accept the model as possible causal explanation of the data. This happens in four out of six cases, except for the cases displayed in rows 1b and 3b, which are rejected.

Rows 1a and 1b concern the same data generated from a nonlinear and acyclic model. We found two different local minima, one of which is accepted (the one more closely resembling the true model), and one is rejected. Even though we learned a causal model with cyclic structure, in the accepted solution, one of the learned causal mechanisms becomes (almost) constant. Rows 3a and 3b show again two different solutions for the same data, now generated from a nonlinear cyclic model. Note that the solution in row 3b could be preferred over that in row 3a based upon its likelihood, but is actually rejected because its estimated noises are highly dependent. Row 4 shows data from a linear, cyclic model, where the ratio of the noise sources equals the ratio of the slopes of the causal mechanisms. This makes this linear model part of the special class of unidentifiable additive noise models. In this case, the MAP estimates for the causal mechanisms are quite different from the true ones.

## 6 Discussion and Conclusion

We have studied a particular class of cyclic causal models given by nonlinear SEMs with additive noise. We have discussed how these models can be interpreted to describe the equilibrium distribution of a dynamic system with noise that is constant in time. We have looked in detail at the bivariate Gaussian-noise case and shown generic identifiability of the causal graph. We have also proposed a method to learn such models from observational data and illustrated it on synthetic data.

Even though we have shown that in this "laboratory setting", the method can be made to work on purely observational data when enough data is available, it includes several assumptions that make it challenging to apply in real-world scenarios. Also, from our experiments, it appears that the method often finds other solutions (local minima of the log likelihood) which differ from the expected true data generating model but which have dependent estimated noises.

Thus there is ample opportunity for future work: For example, improving the robustness of the learning method, and generalizing the results to many variables and non-Gaussian noise.

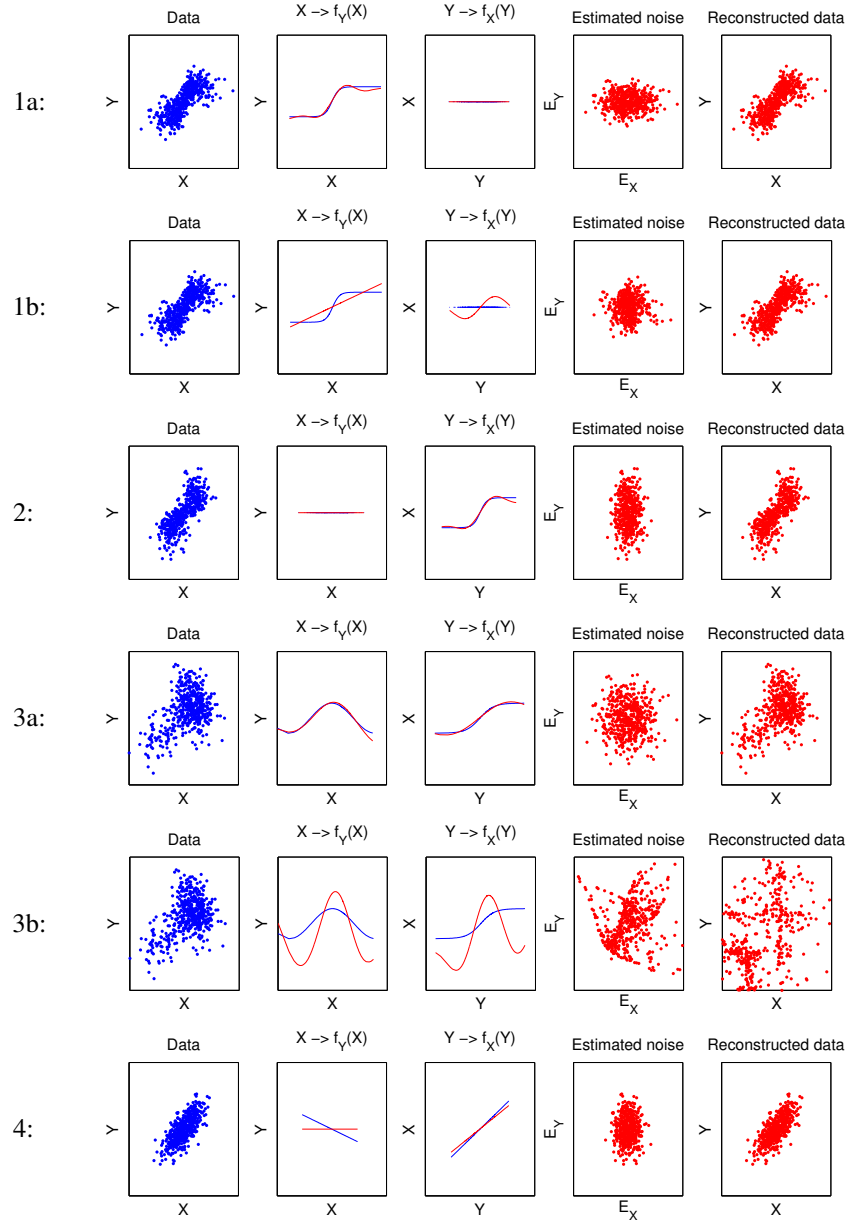

Figure 1: From left to right: observed data pairs $(x, y)$, true (blue) and estimated (red) functions $f_Y$ and $f_X$, respectively, estimated noise values $(e_X, e_Y)$ and reconstructed data $(x, y)$ based on the estimated noise. Rows 1a and 1b show two different solutions (minima of the log likelihood) for the same data, as do rows 3a and 3b. The true models used to generate the data, the $p$-values for independence of the estimated residuals, and the negative log-likelihoods are, from top to bottom:

| # | Identifiable? | Linear? | Cyclic? | $f_Y(x)$ | $f_X(y)$ | $\sigma_X$ | $\sigma_Y$ | $p_{E_X \perp\!\!\!\perp E_Y}$ | $\mathcal{L}$ |
|---|---|---|---|---|---|---|---|---|---|
| 1a | + | − | − | $0.9\tanh(2x)$ | 0 | 1 | 0.5 | 0.76 | $-2.56 \times 10^3$ |
| 1b | + | − | − | $0.9\tanh(2x)$ | 0 | 1 | 0.5 | $7 \times 10^{-3}$ | $-2.51 \times 10^3$ |
| 2 | + | − | − | 0 | $0.9\tanh(2x)$ | 0.5 | 1 | 0.74 | $-2.57 \times 10^3$ |
| 3a | + | − | + | $0.9\cos(x)$ | $0.9\tanh(y)$ | 1 | 1 | 0.78 | $-2.24 \times 10^3$ |
| 3b | + | − | + | $0.9\cos(x)$ | $0.9\tanh(y)$ | 1 | 1 | $3 \times 10^{-58}$ | $-2.26 \times 10^3$ |
| 4 | − | + | + | $-0.4x$ | $0.8y$ | 0.5 | 1 | 0.61 | $-2.73 \times 10^3$ |

## Acknowledgments

We thank Stefan Maubach and Wieb Bosma for their help with the computer algebra. DJ was supported by DFG, the German Research Foundation (SPP 1395). TH and JM were supported by NWO, the Netherlands Organization for Scientific Research (VICI grant 639.023.604 and VENI grant 639.031.036, respectively).

## Footnotes

[1]If some causal mechanism $f_j$ does not depend on one of its parents $i \in \mathrm{pa}(j)$, i.e., if $\frac{\partial f_j}{\partial X_i}(X_{\mathrm{pa}(j)}) = 0$ everywhere, then we discard the edge $i \to j$.

[2]Cyclic additive noise models are also known as "non-recursive" (nonlinear) structural equation models, whereas the acyclic versions are known as "recursive" (nonlinear) SEMs. This terminology is common usage but confusing, as it is precisely in the cyclic case that one needs a recursive procedure to calculate the solutions of equations (2), and not the other way around.

[3]Or similar equations with the roles of $X$ and $Y$ reversed.

[4]Note that if a certain model leads to independent noise terms, then adding more arrows will still allow independent noise terms, by setting some functions to 0—see also Figure 1 below.

## References

[1] J. Pearl. *Causality: Models, Reasoning, and Inference*. Cambridge University Press, 2000.

[2] K. A. Bollen. *Structural Equations with Latent Variables*. John Wiley & Sons, 1989.

[3] P. Spirtes, C. Glymour, and R. Scheines. *Causation, Prediction, and Search*. Springer-Verlag, 1993. (2nd ed. MIT Press 2000).

[4] C.W.J. Granger. Investigating causal relations by econometric models and cross-spectral methods. *Econometrica*, 37:424438, 1969.

[5] N. Friedman, K. Murphy, and S. Russell. Learning the structure of dynamic probabilistic networks. In *Proceedings of the Fourteenth Conference on Uncertainty in Artificial Intelligence (UAI-98)*, pages 139–147, 1998.

[6] P. Spirtes. Directed cyclic graphical representations of feedback models. In *Proceedings of the 11th Conference on Uncertainty in Artificial Intelligence (UAI-95)*, page 491499, 1995.

[7] T. Richardson. A discovery algorithm for directed cyclic graphs. In *Proceedings of the Twelfth Conference on Uncertainty in Artificial Intelligence (UAI-1996)*, 1996.

[8] G. Lacerda, P. Spirtes, J. Ramsey, and P. O. Hoyer. Discovering cyclic causal models by independent components analysis. In *Proceedings of the 24th Conference on Uncertainty in Artificial Intelligence (UAI-2008)*, 2008.

[9] M. Schmidt and K. Murphy. Modeling discrete interventional data using directed cyclic graphical models. In *Proceedings of the 25th Annual Conference on Uncertainty in Artificial Intelligence (UAI-09)*, 2009.

[10] S. Itani, M. Ohannessian, K. Sachs, G. P. Nolan, and M. A. Dahleh. Structure learning in causal cyclic networks. In *JMLR Workshop and Conference Proceedings*, volume 6, page 165176, 2010.

[11] P.O. Hoyer, D.Janzing, J.M.Mooij, J.Peters, and B.Schölkopf. Nonlinear causal discovery with additive noise models. In D. Koller, D. Schuurmans, Y. Bengio, and L. Bottou, editors, *Advances in Neural Information Processing Systems 21 (NIPS*2008)*, pages 689–696, 2009.

[12] Jonas Peters, Joris M. Mooij, Dominik Janzing, and Bernhard Schölkopf. Identifiability of causal graphs using functional models. In *Proceedings of the 27th Annual Conference on Uncertainty in Artificial Intelligence (UAI-11)*, 2011.

[13] K. Zhang and A. Hyvärinen. On the identifiability of the post-nonlinear causal model. In *Proceedings of the 25th Conference on Uncertainty in Artificial Intelligence (UAI-09)*, Montreal, Canada, 2009.

[14] A.D. Polyanin and V.F. Zaitsev. *Handbook of Nonlinear Partial Differential Equations*. Chapman & Hall / CRC, 2004.

[15] A. Gretton, R. Herbrich, A. Smola, O. Bousquet, and B. Schölkopf. Kernel methods for measuring independence. *Journal of Machine Learning Research*, 6:2075–2129, 2005.

